# SEEMORE: A View-Based Approach to 3-D Object Recognition Using Multiple Visual Cues

**Bartlett W. Mel**
Department of Biomedical Engineering
University of Southern California
Los Angeles, CA 90089
mel@quake.usc.edu

## Abstract

A neurally-inspired visual object recognition system is described called SEEMORE, whose goal is to identify common objects from a large known set—independent of 3-D viewing angle, distance, and non-rigid distortion. SEEMORE's database consists of 100 objects that are rigid (shovel), non-rigid (telephone cord), articulated (book), statistical (shrubbery), and complex (photographs of scenes). Recognition results were obtained using a set of 102 color and shape feature channels within a simple feedforward network architecture. In response to a test set of 600 novel test views (6 of each object) presented individually in color video images, SEEMORE identified the object correctly 97% of the time (chance is 1%) using a nearest neighbor classifier. Similar levels of performance were obtained for the subset of 15 non-rigid objects. Generalization behavior reveals emergence of striking natural category structure not explicit in the input feature dimensions.

## 1 INTRODUCTION

In natural contexts, visual object recognition in humans is remarkably fast, reliable, and viewpoint invariant. The present approach to object recognition is "view-based" (e.g. see [Edelman and Bulthoff, 1992]), and has been guided by three main dogmas.

First, the "natural" object recognition problem faced by visual animals involves a large number of objects and scenes, extensive visual experience, and no artificial

distinctions among object classes, such as rigid, non-rigid, articulated, etc.

Second, when an object is recognized in the brain, the "heavy lifting" is done by the first wave of action potentials coursing from the retina to the inferotemporal cortex (IT) over a period of 100 ms [Oram and Perrett, 1992]. The computations carried out during this time can be modeled as a shallow but very wide feedforward network of simple image filtering operations. Shallow means few processing levels, wide means a sparse, high-dimensional representation combining cues from multiple visual submodalities, such as color, texture, and contour [Tanaka et al., 1991].

Third, more complicated processing mechanisms, such as those involving focal attention, segmentation, binding, normalization, mental rotation, dynamic links, parts recognition, etc., may exist and may enhance recognition performance but are not necessary to explain rapid, robust recognition with objects in normal visual situations.

In this vein, the main goal of this project has been to explore the limits of performance of a shallow—but very wide—feedforward network of simple filtering operations for viewpoint-invariant 3-D object recognition, where the filter "channels" themselves have been loosely modeled after the shape- and color-sensitive visual response properties seen in the higher levels of the primate visual system [Tanaka et al., 1991]. Architecturally similar approaches to vision have been most often applied in the domain of optical character recognition [Fukushima et al., 1983, Le Cun et al., 1990]. SEEMORE's architecture is also similar in spirit to the color histogramming approach of [Swain and Ballard, 1991], but includes spatially-structured features that provide also for shape-based generalization.

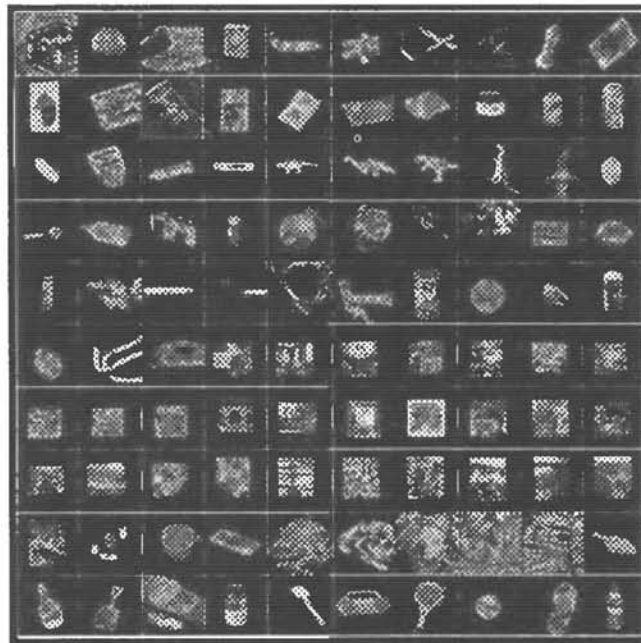

Figure 1: The database includes 100 objects of many different types, including rigid (soup can), non-rigid (necktie), statistical (bunch of grapes), and photographs of complex indoor and outdoor scenes.

## 2   SEEMORE'S VISUAL WORLD

SEEMORE's database contains 100 common 3-D objects and photogaphs of scenes, each represented by a set of pre-segmented color video images (fig. 1). The training set consisted of 12–36 views of each object as follows. For rigid objects, 12 training views were chosen at roughly 60° intervals in depth around the viewing sphere, and each view was then scaled to yield a total of three images at 67%, 100%, and 150%. Image plane orientation was allowed to vary arbitrarily. For non-rigid objects, 12 training views were chosen in random poses.

During a recognition trial, SEEMORE was required to identify novel test images of the database objects. For rigid objects, test images were drawn from the viewpoint interstices of the training set, excluding highly foreshortened views (e.g. bottom of can). Each test view could therefore be presumed to be correctly recognizable, but never closer than roughly 30° in orientation in depth or 22% in scale to the nearest training view of the object, while position and orientation in the image plane could vary arbitrarily. For non-rigid objects, test images consisted of novel random poses. Each test view depicted the isolated object on a smooth background.

### 2.1   FEATURE CHANNELS

SEEMORE's internal representation of a view of an object is encoded by a set of feature channels. The $i$th channel is based on an elemental nonlinear filter $f_i(x, y, \theta_1, \theta_2, \ldots)$, parameterized by position in the visual field and zero or more internal degrees of freedom. Each channel is by design relatively sensitive to changes in the image that are strongly related to object identity, such as the object's shape, color, or texture, while remaining relatively insensitive to changes in the image that are unrelated to object identity, such as are caused by changes in the object's pose. In practice, this invariance is achieved in a straightforward way for each channel by subsampling and summing the output of the elemental channel filter over the entire visual field and one or more of its internal degrees of freedom, giving a channel output $F_i = \sum_{x,y,\theta_1,\ldots} f_i()$. For example, a particular shape-sensitive channel might "look" for the image-plane projections of right-angle corners, over the entire visual field, 360° of rotation in the image plane, 30° of rotation in depth, one octave in scale, and tolerating partial occlusion and/or slight misorientation of the elemental contours that define the right angle. In general, then, $F_i$ may be viewed as a "cell" with a large receptive field whose output is an estimate of the number of occurences of distal feature $i$ in the workspace over a large range of viewing parameters.

SEEMORE's architecture consists of 102 feature channels, whose outputs form an input vector to a nearest-neighbor classifer. Following the design of the individual channels, the channel vector $\mathbf{F} = \{F_1, \ldots F_{102}\}$ is (1) insensitive to changes in image plane position and orientation of the object, (2) modestly sensitive to changes in object scale, orientation in depth, or non-rigid deformation, but (3) highly sensitive to object "quality" as pertains to object identity. Within this representation, total memory storage for all views of an object ranged from 1,224 to 3,672 integers.

As shown in fig. 2, SEEMORE's channels fall into in five groups: (1) 23 color channels, each of which responds to a small blob of color parameterized by "best" hue and saturation, (2) 11 coarse-scale intensity corner channels parameterized by open angle, (3) 12 "blob" features, parameterized by the shape (round and elongated) and

size (small, medium, and large) of bright and dark intensity blobs, (4) 24 contour shape features, including straight angles, curve segments of varying radius, and parallel and oblique line combinations, and (5) 16 shape/texture-related features based on the outputs of Gabor functions at 5 scales and 8 orientations. The implementations of the channel groups were crude, in the interests of achieving a working, multiple-cue system with minimal development time. Images were grabbed using an off-the-shelf Sony S-Video Camcorder and SunVideo digitizing board.

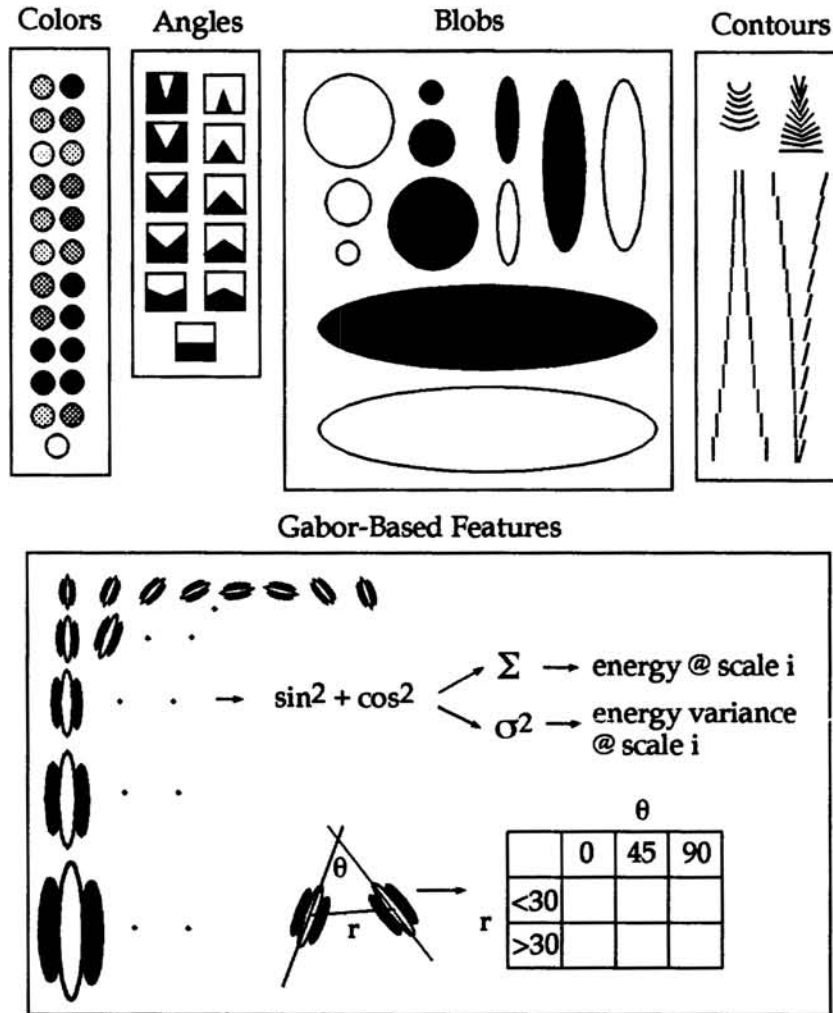

Figure 2: SEEMORE's 102 channels fall into 5 groups, sensitive to (1) colors, (2) intensity corners, (3) circular and elongated intensity blobs, (4) contour shape features, and (5) 16 oriented-energy and relative-orientation features based on the outputs of Gabor functions at several scales and orientations.

## 3  RECOGNITION

SEEMORE's recognition performance was assesed quantitatively as follows. A test set consisting of 600 novel views (100 objects x 6 views) was culled from the database, and presented to SEEMORE for identification. It was noted empirically that a compressive transform on the feature dimensions (histogram values) led to improved classification performance; prior to all learning and recognition operations,

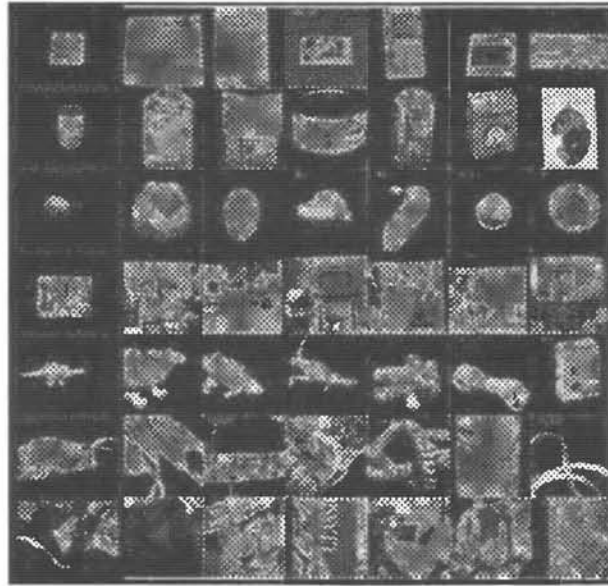

Figure 3: Generalization using only shape-related channels. In each row, a novel test view is shown at far left. The sequence of best matching training views (one per object) is shown to right, in order of decreasing similarity.

therefore, each feature value was replaced by its natural logarithm (0 values were first replaced with a small positive constant to prevent the logarithm from blowing up). For each test view, the city-block distance was computed to every training view in the database and the nearest neighbor was chosen as the best match. The log transform of the feature dimensions thus tied this distance to the ratios of individual feature values in two images rather than their differences.

## 4   RESULTS

Recognition time on a Sparc-20 was 1-2 minutes per view; the bulk of the time was devoted to shape processing, with under 2 seconds required for matching.

Recognition results are reported as the proportion of test views that were correctly classified. Performance using all 102 channels for the 600 novel object views in the intact test set was 96.7%; the chance rate of correct classification was 1%. Across recognition conditions, second-best matches usually accounted for approximately half the errors. Results were broken down in terms of the separate contributions to recognition performance of color-related vs. shape-related feature channels. Performance using only the 23 color-related channels was 87.3%, and using only the 79 shape-related channels was 79.7%. Remarkably, very similar performance figures were obtained for the subset of 90 test views of the non-rigid objects, which included several scarves, a bike chain, necklace, belt, sock, necktie, maple-leaf cluster, bunch of grapes, knit bag, and telephone cord. Thus, a novel random configuration of a telephone cord was as easily recognized as a novel view of a shovel.

## 5   GENERALIZATION BEHAVIOR

Numerical indices of recognition performance are useful, but do not explicitly convey the similarity structure of the underlying feature space. A more qualitative but extremely informative representation of system performance lies in the sequence of images in order of increasing distance from a test view. Records of this kind are shown in fig. 3 for trials in which only shape-related channels were used. In each, a test view is shown at the far left, and the ordered set of nearest neighbors is shown to the right. When a test view's nearest neighbor (second image from left) was not the correct match, the trial was classified as an error.

As shown in row (1), a view of a book is judged most similar to a series of other books (or the bottom of a rectangular cardboard box)—each a view of a rectangular object with high-frequency surface markings. A similar sequence can be seen in subsequent rows for (2) a series of cans, each a right cylinder with detailed surface markings, (3) a series of smooth, not-quite-round objects, (4) a series of photographs of complex scenes, and (5) a series of dinosaurs (followed by a teddy bear). In certain cases, SEEMORE's shape-related similarity metric was more difficult to visually interpret or verbalize (last two rows), or was different from that of a human observer.

## 6   DISCUSSION

The ecology of natural object vision gives rise to an apparent contradiction: (i) generalization in shape-space must in some cases permit an object whose global shape has been grossly perturbed to be matched to itself, such as the various tangled forms of a telephone cord, but (ii) quasi-rigid basic-level shape categories (e.g. chair, shoe, tree) must be preserved as well, and distinguished from each other.

A partial resolution to this conundrum lies in the observation that locally-computed shape statistics are in large part preserved under the global shape deformations that non-rigid common objects (e.g. scarf, bike-chain) typically undergo. A feature-space representation with an emphasis on locally-derived shape channels will therefore exhibit a significant degree of invariance to global nonrigid shape deformations. The definition of shape similarity embodied in the present approach is that two objects are similar if they contain similar profiles (histograms) of their shape measures, which emphasize locality. One way of understanding the emergence of global shape categories, then, such as "book", "can", "dinosaur", etc., is to view each as a set of instances of a single canonical object whose local shape statistics remain quasi-stable as it is warped into various global forms. In many cases, particularly within rigid object categories, exemplars may share longer-range shape statistics as well.

It is useful to consider one further aspect of SEEMORE's shape representation, pertaining to an apparent mismatch between the simplicity of the shape-related feature channels and the complexity of the shape categories that can emerge from them. Specifically, the order of binding of spatial relations within SEEMORE's shape channels is relatively low, i.e. consisting of single simple open or closed curves, or conjunctions of two oriented contours or Gabor patches. The fact that shape categories, such as "photographs of rooms", or "smooth lumpy objects", cluster together in a feature space of such low binding order would therefore at first seem surprising. This phenomenon relates closely to the notion of "wickelfeatures" (see [Rumelhart and McClelland, 1986], ch. 18), in which features (relating to phonemes)

that bind spatial information only locally are nonetheless used to represent global patterns (words) with little or no residual ambiguity.

The presegmentation of objects is a simplifying assumption that is clearly invalid in the real world. The advantage of the assumption from a methodological perspective is that the object similarity structure induced by the feature dimensions can be studied independently from the problem of segmenting or indexing objects imbedded in complex scenes. In continuing work, we are pursuing a leap to sparse very-high-dimensional space (e.g. 10,000 dimensions), whose advantages for classification in the presence of noise (or clutter) have been discussed elsewhere [Kanerva, 1988, Califano and Mohan, 1994].

## Acknowledgements

Thanks to József Fiser for useful discussions and for development of the Gabor-based channel set, to Dan Lipofsky and Scott Dewinter for helping in the construction of the image database, and to Christof Koch for providing support at Caltech where this work was initiated. This work was funded by the Office of Naval Research, and the McDonnell-Pew Foundation.

## References

[Califano and Mohan, 1994] Califano, A. and Mohan, R. (1994). Multidimensional indexing for recognizing visual shapes. *IEEE Trans. on PAMI*, 16:373–392.

[Edelman and Bulthoff, 1992] Edelman, S. and Bulthoff, H. (1992). Orientation dependence in the recognition of familiar and novel views of three-dimensional objects. *Vision Res.*, 32:2385–2400.

[Fukushima et al., 1983] Fukushima, K., Miyake, S., and Ito, T. (1983). Neocognitron: A neural network model for a mechanism of visual pattern recognition. *IEEE Trans. Sys. Man & Cybernetics*, SMC-13:826–834.

[Kanerva, 1988] Kanerva, P. (1988). *Sparse distributed memory*. MIT Press, Cambridge, MA.

[Le Cun et al., 1990] Le Cun, Y., Matan, O., Boser, B., Denker, J., Henderson, D., Howard, R., Hubbard, W., Jackel, L., and Baird, H. (1990). Handwritten zip code recognition with multilayer networks. In *Proc. of the 10th Int. Conf. on Patt. Rec.* IEEE Computer Science Press.

[Oram and Perrett, 1992] Oram, M. and Perrett, D. (1992). Time course of neural responses discriminating different views of the face and head. *J. Neurophysiol.*, 68(1):70–84.

[Rumelhart and McClelland, 1986] Rumelhart, D. and McClelland, J. (1986). *Parallel distributed processing*. MIT Press, Cambridge, Massachusetts.

[Swain and Ballard, 1991] Swain, M. and Ballard, D. (1991). Color indexing. *Int. J. Computer Vision*, 7:11–32.

[Tanaka et al., 1991] Tanaka, K., Saito, H., Fukada, Y., and Moriya, M. (1991). Coding visual images of objects in the inferotemporal cortex of the macaque monkey. *J. Neurophysiol.*, 66:170–189.
